# Transfer learning for text classification

**Chuong B. Do**
Computer Science Department
Stanford University
Stanford, CA 94305

**Andrew Y. Ng**
Computer Science Department
Stanford University
Stanford, CA 94305

## Abstract

Linear text classification algorithms work by computing an inner product between a test document vector and a parameter vector. In many such algorithms, including naive Bayes and most TFIDF variants, the parameters are determined by some simple, closed-form, function of training set statistics; we call this mapping mapping from statistics to parameters, the *parameter function.* Much research in text classification over the last few decades has consisted of manual efforts to identify better parameter functions. In this paper, we propose an algorithm for automatically *learning* this function from related classification problems. The parameter function found by our algorithm then defines a *new learning algorithm* for text classification, which we can apply to novel classification tasks. We find that our learned classifier outperforms existing methods on a variety of multiclass text classification tasks.

## 1  Introduction

In the multiclass text classification task, we are given a training set of documents, each labeled as belonging to one of $K$ disjoint classes, and a new unlabeled test document. Using the training set as a guide, we must predict the most likely class for the test document. "Bag-of-words" linear text classifiers represent a document as a vector $\mathbf{x}$ of word counts, and predict the class whose score (a linear function of $\mathbf{x}$) is highest, i.e., $\arg\max_{k\in\{1,...,K\}} \sum_{i=1}^{n} \theta_{ki}x_i$. Choosing parameters $\{\theta_{ki}\}$ which give high classification accuracy on test data, thus, is the main challenge for linear text classification algorithms.

In this paper, we focus on linear text classification algorithms in which the parameters are pre-specified functions of training set statistics; that is, each $\theta_{ki}$ is a function $\theta_{ki} := g(\mathbf{u}_{ki})$ of some *fixed statistics* $\mathbf{u}_{ki}$ of the training set. Unlike discriminative learning methods, such as logistic regression [1] or support vector machines (SVMs) [2], which use numerical optimization to pick parameters, the learners we consider perform no optimization. Rather, in our technique, parameter learning involves tabulating statistics vectors $\{\mathbf{u}_{ki}\}$ and applying the closed-form function $g$ to obtain parameters. We refer to $g$, this mapping from statistics to parameters, as the *parameter function*.

Many common text classification methods—including the multinomial and multivariate Bernoulli event models for naive Bayes [3], the vector space-based TFIDF classifier [4], and its probabilistic variant, PrTFIDF [5]—belong to this class of algorithms. Here, picking a good text classifier from this class is equivalent to finding the right parameter function for the available statistics.

In practice, researchers often develop text classification algorithms by trial-and-error, guided by empirical testing on real-world classification tasks (cf. [6, 7]). Indeed, one could

argue that much of the 30-year history of information retrieval has consisted of manually trying TFIDF formula variants (i.e. adjusting the parameter function $g$) to optimize performance [8]. Even though this heuristic process can often lead to good parameter functions, such a laborious task requires much human ingenuity, and risks failing to find algorithm variations not considered by the designer.

In this paper, we consider the task of *automatically* learning a parameter function $g$ for text classification. Given a set of example text classification problems, we wish to "meta-learn" a new *learning algorithm* (as specified by the parameter function $g$), which may then be applied new classification problems. The meta-learning technique we propose, which leverages data from a variety of related classification tasks to obtain a good classifier for new tasks, is thus an instance of *transfer learning*; specifically, our framework automates the process of finding a good parameter function for text classifiers, replacing hours of hand-tweaking with a straightforward, globally-convergent, convex optimization problem.

Our experiments demonstrate the effectiveness of learning classifier forms. In low training data classification tasks, the learning algorithm given by our automatically learned parameter function consistently outperforms human-designed parameter functions based on naive Bayes and TFIDF, as well as existing discriminative learning approaches.

## 2 Preliminaries

Let $\mathcal{V} = \{w_1, \ldots, w_n\}$ be a fixed vocabulary of words, and let $\mathcal{X} = \mathbb{Z}^n$ and $\mathcal{Y} = \{1, \ldots, K\}$ be the input and output spaces for our classification problem. A *labeled document* is a pair $(\mathbf{x}, y) \in \mathcal{X} \times \mathcal{Y}$, where $\mathbf{x}$ is an $n$-dimensional vector with $x_i$ indicating the number of occurrences of word $w_i$ in the document, and $y$ is the document's class label. A *classification problem* is a tuple $\langle \mathcal{D}, S, (\mathbf{x}_{\text{test}}, y_{\text{test}}) \rangle$, where $\mathcal{D}$ is a distribution over $\mathcal{X} \times \mathcal{Y}$, $S = \{(\mathbf{x}_i, y_i)\}_{i=1}^M$ is a set of $M$ training examples, $(\mathbf{x}_{\text{test}}, y_{\text{test}})$ is a single test example, and all $M + 1$ examples are drawn iid from $\mathcal{D}$. Given a training set $S$ and a test input vector $\mathbf{x}_{\text{test}}$, we must predict the value of the test class label $y_{\text{test}}$.

In linear classification algorithms, we evaluate the score $f_k(\mathbf{x}_{\text{test}}) := \sum_i \theta_{ki} \mathbf{x}_{\text{test}\,i}$ for assigning $\mathbf{x}_{\text{test}}$ to each class $k \in \{1, \ldots, K\}$ and pick the class $y = \arg\max_k f_k(\mathbf{x}_{\text{test}})$ with the highest score. In our meta-learning setting, we define each $\theta_{ki}$ as the component-wise evaluation of the parameter function $g$ on some vector of training set statistics $\mathbf{u}_{ki}$:

$$\begin{bmatrix} \theta_{k1} \\ \theta_{k2} \\ \vdots \\ \theta_{kn} \end{bmatrix} := \begin{bmatrix} g(\mathbf{u}_{k1}) \\ g(\mathbf{u}_{k2}) \\ \vdots \\ g(\mathbf{u}_{kn}) \end{bmatrix}. \tag{1}$$

Here, each $\mathbf{u}_{ki} \in \mathbb{R}^q$ ($k = 1, \ldots, K$, $i = 1, \ldots, n$) is a vector whose components are computed from the training set $S$ (we will provide specific examples later). Furthermore, $g : \mathbb{R}^q \to \mathbb{R}$ is the *parameter function* mapping from $\mathbf{u}_{ki}$ to its corresponding parameter $\theta_{ki}$. To illustrate these definitions, we show that two specific cases of the naive Bayes and TFIDF classification methods belong to the class of algorithms described above.

**Naive Bayes:** In the multinomial variant of the naive Bayes classification algorithm,[1] the score for assigning a document $\mathbf{x}$ to class $k$ is

$$f_k^{\text{NB}}(\mathbf{x}) := \log \hat{p}(y = k) + \sum_{i=1}^n x_i \log \hat{p}(w_i \mid y = k). \tag{2}$$

The first term, $\hat{p}(y = k)$, corresponds to a "prior" over document classes, and the second term, $\hat{p}(w_i \mid y = k)$, is the (smoothed) relative frequency of word

$w_i$ in training documents of class $k$. For balanced training sets, the first term is irrelevant. Therefore, we have $f_k^{\mathrm{NB}}(\mathbf{x}) = \sum_i \theta_{ki} x_i$ where $\theta_{ki} = g_{\mathrm{NB}}(\mathbf{u}_{ki})$,

$$
\mathbf{u}_{ki} := \begin{bmatrix} u_{ki1} \\ u_{ki2} \\ u_{ki3} \\ u_{ki4} \\ u_{ki5} \end{bmatrix} = \begin{bmatrix} \text{number of times } w_i \text{ appears in documents of class } k \\ \text{number of documents of class } k \text{ containing } w_i \\ \text{total number of words in documents of class } k \\ \text{total number of documents of class } k \\ \text{total number of documents} \end{bmatrix}, \quad (3)
$$

and

$$
g_{\mathrm{NB}}(\mathbf{u}_{ki}) := \log \frac{u_{ki1} + \varepsilon}{u_{ki3} + n\varepsilon} \tag{4}
$$

where $\varepsilon$ is a smoothing parameter. ($\varepsilon = 1$ gives Laplace smoothing.)

**TFIDF:** In the unnormalized TFIDF classifier, the score for assigning $\mathbf{x}$ to class $k$ is

$$
f_k^{\mathrm{TFIDF}}(\mathbf{x}) := \sum_{i=1}^n \left( \overline{x}_i|_{y=k} \cdot \log \frac{1}{\hat{p}(x_i > 0)} \right) \left( x_i \cdot \log \frac{1}{\hat{p}(x_i > 0)} \right), \tag{5}
$$

where $\overline{x}_i|_{y=k}$ (sometimes called the average term frequency of $w_i$) is the average $i$th component of all document vectors of class $k$, and $\hat{p}(x_i > 0)$ (sometimes called the document frequency of $w_i$) is the proportion of all documents containing $w_i$.[2] As before, we write $f_k^{\mathrm{TFIDF}}(\mathbf{x}) = \sum_i \theta_{ki} x_i$ with $\theta_{ki} = g_{\mathrm{TFIDF}}(\mathbf{u}_{ki})$. The statistics vector is again defined as in (3), but this time,

$$
g_{\mathrm{TFIDF}}(\mathbf{u}_{ki}) := \frac{u_{ki1}}{u_{ki4}} \left( \log \frac{u_{ki5}}{u_{ki2}} \right)^2. \tag{6}
$$

Space constraints preclude a detailed discussion, but many other classification algorithms can similarly be expressed in this framework, using other definitions of the statistics vectors $\{\mathbf{u}_{ki}\}$. These include most other variants of TFIDF based on different TF and IDF terms [7], PrTFIDF [5], and various heuristically modified versions of naive Bayes [6].

## 3   Learning the parameter function

In the last section, we gave two examples of algorithms that obtain their parameters $\theta_{ki}$ by applying a function $g$ to a statistics vector $\mathbf{u}_{ki}$. In each case, the parameter function was hand-designed, either from probabilistic (in the case of naive Bayes [3]) or geometric (in the case of TFIDF [4]) considerations. We now consider the problem of automatically learning a parameter function from example classification tasks. In the sequel, we assume fixed statistics vectors $\{\mathbf{u}_{ki}\}$ and focus on finding an optimal parameter function $g$.

In the standard supervised learning setting, we are given a training set of examples sampled from some unknown distribution $\mathcal{D}$, and our goal is to use the training set to make a prediction on a new test example also sampled from $\mathcal{D}$. By using the training examples to understand the statistical regularities in $\mathcal{D}$, we hope to predict $y_{\text{test}}$ from $\mathbf{x}_{\text{test}}$ with low error.

Analogously, the problem of meta-learning $g$ is again a supervised learning task; here, however, the training "examples" are now classification problems sampled from a distribution $\mathscr{D}$ *over classification problems*.[3] By seeing many instances of text classification problems

drawn from $\mathscr{D}$, we hope to learn a parameter function $g$ that exploits the statistical regularities in problems from $\mathscr{D}$. Formally, let $\mathscr{S} = \{\langle \mathcal{D}^{(j)}, S^{(j)}, (\mathbf{x}^{(j)}, y^{(j)}) \rangle\}_{j=1}^{m}$ be a collection of $m$ classification problems sampled iid from $\mathscr{D}$. For a new, test classification problem $\langle \mathcal{D}_{\text{test}}, S_{\text{test}}, (\mathbf{x}_{\text{test}}, y_{\text{test}}) \rangle$ sampled independently from $\mathscr{D}$, we desire that our learned $g$ correctly classify $\mathbf{x}_{\text{test}}$ with high probability.

To achieve our goal, we first restrict our attention to parameter functions $g$ that are *linear* in their inputs. Using the linearity assumption, we pose a convex optimization problem for finding a parameter function $g$ that achieves small loss on test examples in the training collection. Finally, we generalize our method to the non-parametric setting via the "kernel trick," thus allowing us to learn complex, highly non-linear functions of the input statistics.

### 3.1 Softmax learning

Recall that in *softmax regression*, the class probabilities $p(y \mid \mathbf{x})$ are modeled as

$$p(y = k \mid \mathbf{x}; \{\theta_{ki}\}) := \frac{\exp(\sum_i \theta_{ki} x_i)}{\sum_{k'} \exp(\sum_i \theta_{k'i} x_i)}, \qquad k = 1, \ldots, K, \tag{7}$$

where the parameters $\{\theta_{ki}\}$ are learned from the training data $S$ by maximizing the conditional log likelihood of the data. In this approach, a total of $Kn$ parameters are trained jointly using numerical optimization. Here, we consider an alternative approach in which each of the $Kn$ parameters is some function of the prespecified statistics vectors; in particular, $\theta_{ki} := g(\mathbf{u}_{ki})$. Our goal is to learn an appropriate $g$.

To pose our optimization problem, we start by learning the *linear form* $g(\mathbf{u}_{ki}) = \boldsymbol{\beta}^T \mathbf{u}_{ki}$. Under this parameterization, the conditional likelihood of an example $(\mathbf{x}, y)$ is

$$p(y = k \mid \mathbf{x}; \boldsymbol{\beta}) = \frac{\exp(\sum_i \boldsymbol{\beta}^T \mathbf{u}_{ki} x_i)}{\sum_{k'} \exp(\sum_i \boldsymbol{\beta}^T \mathbf{u}_{k'i} x_i)}, \qquad k = 1, \ldots, K. \tag{8}$$

In this setup, one natural approach for learning a linear function $g$ is to maximize the (regularized) conditional log likelihood $\ell(\boldsymbol{\beta} : \mathscr{S})$ for the entire collection $\mathscr{S}$:

$$\begin{aligned}
\ell(\boldsymbol{\beta} : \mathscr{S}) &:= \sum_{j=1}^{m} \log p(y^{(j)} \mid \mathbf{x}^{(j)}; \boldsymbol{\beta}) - C||\boldsymbol{\beta}||^2 \\
&= \sum_{j=1}^{m} \log \left( \frac{\exp\left(\boldsymbol{\beta}^T \sum_i \mathbf{u}_{y^{(j)}i}^{(j)} x_i^{(j)}\right)}{\sum_k \exp\left(\boldsymbol{\beta}^T \sum_i \mathbf{u}_{ki}^{(j)} x_i^{(j)}\right)} \right) - C||\boldsymbol{\beta}||^2.
\end{aligned} \tag{9}$$

In (9), the latter term corresponds to a Gaussian prior on the parameters $\boldsymbol{\beta}$, which provides a means for controlling the complexity of the learned parameter function $g$. The maximization of (9) is similar to softmax regression training except that here, instead of optimizing over the parameters $\{\theta_{ki}\}$ directly, we optimize over the choice of $\boldsymbol{\beta}$.

### 3.2 Nonparametric function learning

In this section, we generalize the technique of the previous section to nonlinear $g$. By the Representer Theorem [10], there exists a maximizing solution to (9) for which the optimal parameter vector $\boldsymbol{\beta}^*$ is a linear combination of training set statistics:

$$\boldsymbol{\beta}^* = \sum_{j=1}^{m} \sum_k \alpha_{jk}^* \sum_i \mathbf{u}_{ki}^{(j)} x_i^{(j)}. \tag{10}$$

From this, we reparameterize the original optimization over $\boldsymbol{\beta}$ in (9) as an equivalent optimization over training example weights $\{\alpha_{jk}\}$. For notational convenience, let

$$\mathcal{K}(j, j', k, k') := \sum_i \sum_{i'} x_i^{(j)} x_{i'}^{(j')} (\mathbf{u}_{ki}^{(j)})^T \mathbf{u}_{k'i'}^{(j')}. \tag{11}$$

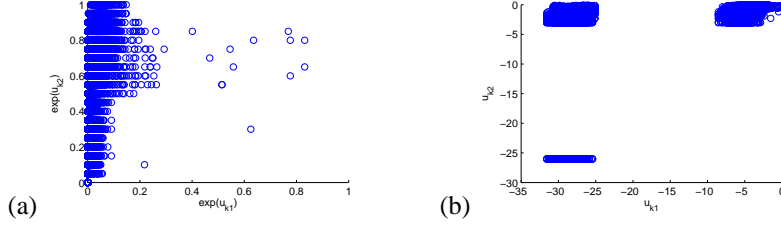

(a)                                                    (b)

Figure 1: Distribution of unnormalized $\mathbf{u}_{ki}$ vectors in dmoz data (a) with and (b) without applying the log transformation in (15). In principle, one could alternatively use a feature vector representation using these frequencies directly, as in (a). However, applying the log transformation yields a feature space with fewer isolated points in $\mathbb{R}^2$, as in (b). When using the Gaussian kernel, a feature space with few isolated points is important as the topology of the feature space establishes locality of influence for support vectors.

Substituting (10) and (11) into (9), we obtain

$$\ell(\{\alpha_{jk}\} : \mathscr{S}) := \sum_{j'=1}^{m} \log \left( \frac{\exp\left(\sum_{j=1}^{m}\sum_{k} \alpha_{jk}\mathcal{K}(j, j', k, y^{(j')})\right)}{\sum_{k'} \exp\left(\sum_{j=1}^{m}\sum_{k} \alpha_{jk}\mathcal{K}(j, j', k, k')\right)} \right)$$
$$- C \sum_{j=1}^{m}\sum_{j'=1}^{m}\sum_{k}\sum_{k'} \alpha_{jk}\alpha_{j'k'}\mathcal{K}(j, j', k, k'). \tag{12}$$

Note that (12) is concave and differentiable, so we can train the model using any standard numerical gradient optimization procedure, such as conjugate gradient or L-BFGS [11].

The assumption that $g$ is a linear function of $\mathbf{u}_{ki}$, however, places a severe restriction on the class of learnable parameter functions. Noting that the statistics vectors appear only as an inner product in (11), we apply the "kernel trick" to obtain

$$\mathcal{K}(j, j', k, k') := \sum_{i}\sum_{i'} x_i^{(j)} x_{i'}^{(j')} K(\mathbf{u}_{ki}^{(j)}, \mathbf{u}_{k'i'}^{(j')}), \tag{13}$$

where the kernel function $K(\mathbf{u}, \mathbf{v}) = \langle \Phi(\mathbf{u}), \Phi(\mathbf{v}) \rangle$ defines the inner product of some high-dimensional mapping $\Phi(\cdot)$ of its inputs.[4] In particular, choosing a Gaussian (RBF) kernel, $K(\mathbf{u}, \mathbf{v}) := \exp(-\gamma||\mathbf{u} - \mathbf{v}||^2)$, gives a *non-parametric* representation for $g$:

$$g(\mathbf{u}_{ki}) = \boldsymbol{\beta}^T \Phi(\mathbf{u}_{ki}) = \sum_{j=1}^{m}\sum_{k}\sum_{i} \alpha_{jk} x_i^{(j)} \exp(-\gamma||\mathbf{u}_{ki}^{(j)} - \mathbf{u}_{ki}||^2). \tag{14}$$

Thus, $g(\mathbf{u}_{ki})$ is a weighted combination of the values $\{\alpha_{jk}x_i^{(j)}\}$, where the weights depend exponentially on the squared $\ell_2$-distance of $\mathbf{u}_{ki}$ to each of the statistics vectors $\{\mathbf{u}_{ki}^{(j)}\}$. As a result, we can approximate any sufficiently smooth bounded function of $\mathbf{u}$ arbitrarily well, given sufficiently many training classification problems.

## 4 Experiments

To validate our method, we evaluated its ability to learn parameter functions on a variety of email and webpage classification tasks in which the number of classes, $K$, was large ($K = 10$), and the number of number of training examples per class, $m/K$, was small ($m/K = 2$). We used the dmoz Open Directory Project hierarchy,[5] the 20 Newsgroups dataset,[6] the Reuters-21578 dataset,[7] and the Industry Sector dataset[8].

Table 1: Test set accuracy on dmoz categories. Columns 2-4 give the proportion of correct classifications using non-discriminative methods: the learned $g$, Naive Bayes, and TFIDF, respectively. Columns 5-7 give the corresponding values for the discriminative methods: softmax regression, 1-vs-all SVMs, and multiclass SVMs. The best accuracy in each row is shown in bold.

| Category | $g$ | $g_{\text{NB}}$ | $g_{\text{TFIDF}}$ | softmax | 1VA-SVM | MC-SVM |
|---|---|---|---|---|---|---|
| Arts | **0.421** | 0.296 | 0.286 | 0.352 | 0.203 | 0.367 |
| Business | **0.456** | 0.283 | 0.286 | 0.336 | 0.233 | 0.340 |
| Computers | **0.467** | 0.304 | 0.327 | 0.344 | 0.217 | 0.387 |
| Games | **0.411** | 0.288 | 0.240 | 0.279 | 0.240 | 0.330 |
| Health | **0.479** | 0.282 | 0.337 | 0.382 | 0.213 | 0.337 |
| Home | **0.640** | 0.470 | 0.454 | 0.501 | 0.333 | 0.440 |
| Kids and Teens | **0.252** | 0.205 | 0.142 | 0.202 | 0.173 | 0.167 |
| News | 0.349 | 0.222 | 0.212 | 0.382 | 0.270 | **0.397** |
| Recreation | **0.663** | 0.487 | 0.529 | 0.477 | 0.353 | 0.590 |
| Reference | **0.635** | 0.415 | 0.458 | 0.602 | 0.383 | 0.543 |
| Regional | **0.438** | 0.268 | 0.258 | 0.329 | 0.260 | 0.357 |
| Science | **0.363** | 0.256 | 0.246 | 0.353 | 0.223 | 0.340 |
| Shopping | **0.612** | 0.456 | 0.556 | 0.483 | 0.373 | 0.550 |
| Society | **0.435** | 0.308 | 0.285 | 0.379 | 0.213 | 0.377 |
| Sports | **0.619** | 0.432 | 0.285 | 0.507 | 0.267 | 0.527 |
| World | **0.531** | 0.491 | 0.352 | 0.329 | 0.277 | 0.303 |
| Average | **0.486** | 0.341 | 0.328 | 0.390 | 0.264 | 0.397 |

The dmoz project is a hierarchical collection of webpage links organized by subject matter. The top level of the hierarchy consists of 16 major categories, each of which contains several subcategories. To perform cross-validated testing, we obtained classification problems from each of the top-level categories by retrieving webpages from each of their respective subcategories. For the 20 Newsgroups, Reuters-21578, and Industry Sector datasets, we performed similar preprocessing.[9] Given a dataset of documents, we sampled 10-class 2-training-examples-per-class classification problems by randomly selecting 10 different classes within the dataset, picking 2 training examples within each class, and choosing one test example from a randomly chosen class.

## 4.1 Choice of features

Theoretically, for the method described in this paper, any sufficiently rich set of features could be used to learn a parameter function for classification. For simplicity, we reduced the feature vector in (3) to the following two-dimensional representation,[10]

$$\mathbf{u}_{ki} = \begin{bmatrix} \log(\text{proportion of } w_i \text{ among words from documents of class } k) \\ \log(\text{proportion of documents containing } w_i) \end{bmatrix}. \quad (15)$$

Note that up to the log transformation, the components of $\mathbf{u}_{ki}$ correspond to the relative term frequency and document frequency of a word relative to class $k$ (see Figure 1).

## 4.2 Generalization performance

We tested our meta-learning algorithm on classification problems taken from each of the 16 top-level dmoz categories. For each top-level category, we built a collection of 300 classification problems from that category; results reported here are averages over these

Table 2: Cross corpora classification accuracy, using classifiers trained on each of the four corpora. The best accuracy in each row is shown in bold.

| Dataset | $g_{\text{dmoz}}$ | $g_{\text{news}}$ | $g_{\text{reut}}$ | $g_{\text{indu}}$ | $g_{\text{NB}}$ | $g_{\text{TFIDF}}$ | softmax | 1VA-SVM | MC-SVM |
|---|---|---|---|---|---|---|---|---|---|
| dmoz | n/a | 0.471 | **0.475** | 0.473 | 0.365 | 0.352 | 0.381 | 0.283 | 0.412 |
| 20 Newsgroups | 0.369 | n/a | **0.371** | 0.369 | 0.223 | 0.184 | 0.217 | 0.206 | 0.248 |
| Reuters-21578 | 0.567 | 0.567 | n/a | **0.619** | 0.463 | 0.475 | 0.463 | 0.308 | 0.481 |
| Industry Sector | 0.438 | **0.459** | 0.446 | n/a | 0.374 | 0.274 | 0.376 | 0.271 | 0.375 |

problems. To assess the accuracy of our meta-learning algorithm for a particular test category, we used the $g$ learned from a set of 450 classification problems drawn from the *other* 15 top-level categories.[11] This ensured no overlap of training and testing data. In 15 out of 16 categories, the learned parameter function $g$ outperforms naive Bayes and TFIDF in addition to the discriminative methods we tested (softmax regression, 1-vs-all SVMs [12], and multiclass SVMs [13][12]; see Table 1).[13]

Next, we assessed the ability of $g$ to transfer across even more dissimilar corpora. Here, for each of the four corpora (dmoz, 20 Newsgroups, Reuters-21578, Industry Sector), we constructed independent training and testing datasets of 480 random classification problems. After training separate classifiers ($g_{\text{dmoz}}$, $g_{\text{news}}$, $g_{\text{reut}}$, and $g_{\text{indu}}$) using data from each of the four corpora, we tested the performance of each learned classifier on the remaining three corpora (see Table 2). Again, the learned parameter functions compare favorably to the other methods. Moreover, these tests show that a single parameter function may give an accurate classification algorithm for many different corpora, demonstrating the effectiveness of our approach for achieving transfer across related learning tasks.

## 5   Discussion and Related Work

In this paper, we presented an algorithm based on softmax regression for learning a parameter function $g$ from example classification problems. Once learned, $g$ defines a new learning algorithm that can be applied to novel classification tasks.

Another approach for learning $g$ is to modify the multiclass support vector machine formulation of Crammer and Singer [13] in a manner analogous to the modification of softmax regression in Section 3.1, giving the following quadratic program:

$$\begin{aligned}
&\underset{\boldsymbol{\beta}\in\mathbb{R}^n,\,\boldsymbol{\xi}\in\mathbb{R}^m}{\text{minimize}} && \tfrac{1}{2}||\boldsymbol{\beta}||^2 + C\sum_j \xi_j \\
&\text{subject to} && \boldsymbol{\beta}^T \sum_i x_i^{(j)}(\mathbf{u}_{y^{(j)}i}^{(j)} - \mathbf{u}_{ki}^{(j)}) \geq \mathbf{I}_{\{k\neq y^{(j)}\}} - \xi_j, \quad \forall k, \forall j.
\end{aligned}$$

As usual, taking the dual leads naturally to an SMO-like procedure for optimization. We implemented this method and found that the learned $g$, like in the softmax formulation, outperforms naive Bayes, TFIDF, and the other discriminative methods.

The techniques described in this paper give one approach for achieving *inductive transfer* in classifier design—using labeled data from related example classification problems to solve a particular classification problem [16, 17]. Bennett et al. [18] also consider the issue of knowledge transfer in text classification in the context of ensemble classifiers, and propose a system for using related classification problems to learn the reliability of individual classifiers within the ensemble. Unlike their approach, which attempts to meta-learn *properties*

of algorithms, our method uses meta-learning to *construct* a new classification algorithm. Though not directly applied to text classification, Teevan and Karger [19] consider the problem of automatically learning term distributions for use in information retrieval.

Finally, Thrun and O'Sullivan [20] consider the task of classification in a mobile robot domain. In this work, the authors describe a task-clustering (TC) algorithm in which learning tasks are grouped via a nearest neighbors algorithm, as a means of facilitating knowledge transfer. A similar concept is implicit in the kernelized parameter function learned by our algorithm, where the Gaussian kernel facilitates transfer between similar statistics vectors.

## Acknowledgments

We thank David Vickrey and Pieter Abbeel for useful discussions, and the anonymous referees for helpful comments. CBD was supported by an NDSEG fellowship. This work was supported by DARPA under contract number FA8750-05-2-0249.

## Footnotes

[1]Despite naive Bayes' overly strong independence assumptions and thus its shortcomings as a probabilistic *model* for text documents, we can nonetheless view naive Bayes as simply an *algorithm* which makes predictions by computing certain functions of the training set. This view has proved useful for analysis of naive Bayes even when none of its probabilistic assumptions hold [9]; here, we adopt this view, without attaching any particular probabilistic meaning to the empirical frequencies $\hat{p}(\cdot)$ that happen to be computed by the algorithm.

[2]Note that (5) implicitly defines $f_k^{\mathrm{TFIDF}}(\mathbf{x})$ as a dot product of two vectors, each of whose components consist of a product of two terms. In the normalized TFIDF classifier, both vectors are normalized to unit length before computing the dot product, a modification that makes the algorithm more stable for documents of varying length. This too can be represented within our framework by considering appropriately normalized statistics vectors.

[3]Note that in our meta-learning problem, the output of our algorithm is a parameter function $g$ mapping statistics to parameters. Our training data, however, do not explicitly indicate the best parameter function $g^*$ for each example classification problem. Effectively then, in the meta-learning task, the central problem is to fit $g$ to some *unseen* $g^*$, based on test examples in each training classification problem.

[4]Note also that as a consequence of our kernelization, $\mathcal{K}$ itself can be considered a 'kernel' between all statistics vectors from two entire documents.

[5]http://www.dmoz.org

[6]http://kdd.ics.uci.edu/databases/20newsgroups/20newsgroups.tar.gz

[7]http://www.daviddlewis.com/resources/testcollections/reuters21578/reuters21578.tar.gz

[8]http://www.cs.umass.edu/~mccallum/data/sector.tar.gz

[9]For the Reuters data, we associated each article with its hand-annotated 'topic' label and discarded any articles with more than one topic annotation. For each dataset, we discarded all categories with fewer than 50 examples, and selected a 500-word vocabulary based on information gain.

[10]Features were rescaled to have zero mean and unit variance over the training set.

[11]For each execution of the learning algorithm, $(C, \gamma)$ parameters were determined via grid search using a small holdout set of 160 classification problems. The same holdout set was used to select regularization parameters for the discriminative learning algorithms.

[12]We used LIBSVM [14] to assess 1VA-SVMs and SVM-Light [15] for multiclass SVMs.

[13]For larger values of $m/K$ (e.g. $m/K = 10$), softmax and multiclass SVMs consistently outperform naive Bayes and TFIDF; nevertheless, the learned $g$ achieves a performance on par with discriminative methods, despite being constrained to parameters which are explicit functions of training data statistics. This result is consistent with a previous study in which a heuristically hand-tuned version of Naive Bayes attained near-SVM text classification performance for large datasets [6].

## References

[1] K. Nigam, J. Lafferty, and A. McCallum. Using maximum entropy for text classification. In *IJCAI-99 Workshop on Machine Learning for Information Filtering*, pages 61–67, 1999.

[2] T. Joachims. Text categorization with support vector machines: Learning with many relevant features. In *Machine Learning: ECML-98*, pages 137–142, 1998.

[3] A. McCallum and K. Nigam. A comparison of event models for Naive Bayes text classification. In *AAAI-98 Workshop on Learning for Text Categorization*, 1998.

[4] G. Salton and C. Buckley. Term weighting approaches in automatic text retrieval. *Information Processing and Management*, 29(5):513–523, 1988.

[5] T. Joachims. A probabilistic analysis of the Rocchio algorithm with TFIDF for text categorization. In *Proceedings of ICML-97*, pages 143–151, 1997.

[6] J. D. Rennie, L. Shih, J. Teevan, and D. R. Karger. Tackling the poor assumptions of naive Bayes text classifiers. In *ICML*, pages 616–623, 2003.

[7] A. Moffat and J. Zobel. Exploring the similarity space. In *ACM SIGIR Forum 32*, 1998.

[8] C. Manning and H. Schutze. Foundations of statistical natural language processing, 1999.

[9] A. Ng and M. Jordan. On discriminative vs. generative classifiers: a comparison of logistic regression and naive Bayes. In *NIPS 14*, 2002.

[10] G. Kimeldorf and G. Wahba. Some results on Tchebycheffian spline functions. *J. Math. Anal. Appl.*, 33:82–95, 1971.

[11] J. Nocedal and S. J. Wright. *Numerical Optimization*. Springer, 1999.

[12] R. Rifkin and A. Klautau. In defense of one-vs-all classification. *J. Mach. Learn. Res.*, 5:101–141, 2004.

[13] K. Crammer and Y. Singer. On the algorithmic implementation of multiclass kernel-based vector machines. *J. Mach. Learn. Res.*, 2:265–292, 2001.

[14] C-C. Chang and C-J. Lin. *LIBSVM: a library for support vector machines*, 2001. Software available at `http://www.csie.ntu.edu.tw/~cjlin/libsvm`.

[15] T. Joachims. Making large-scale support vector machine learning practical. In *Advances in Kernel Methods: Support Vector Machines*. MIT Press, Cambridge, MA, 1998.

[16] S. Thrun. Lifelong learning: A case study. CMU tech report CS-95-208, 1995.

[17] R. Caruana. Multitask learning. *Machine Learning*, 28(1):41–75, 1997.

[18] P. N. Bennett, S. T. Dumais, and E. Horvitz. Inductive transfer for text classification using generalized reliability indicators. In *Proceedings of ICML Workshop on The Continuum from Labeled to Unlabeled Data in Machine Learning and Data Mining*, 2003.

[19] J. Teevan and D. R. Karger. Empirical development of an exponential probabilistic model for text retrieval: Using textual analysis to build a better model. In *SIGIR '03*, 2003.

[20] S. Thrun and J. O'Sullivan. Discovering structure in multiple learning tasks: The TC algorithm. In *International Conference on Machine Learning*, pages 489–497, 1996.
